# Analytical Results for the Error in Filtering of Gaussian Processes

**Alex Susemihl**
Bernstein Center for Computational Neuroscience Berlin,Technische Universität Berlin
alex.susemihl@bccn-berlin.de

**Ron Meir**
Department of Eletrical Engineering, Technion, Haifa
rmeir@ee.technion.ac.il

**Manfred Opper**
Bernstein Center for Computational Neuroscience Berlin, Technische Universität Berlin
opperm@cs.tu-berlin.de

## Abstract

Bayesian filtering of stochastic stimuli has received a great deal of attention recently. It has been applied to describe the way in which biological systems dynamically represent and make decisions about the environment. There have been no exact results for the error in the biologically plausible setting of inference on point process, however. We present an exact analysis of the evolution of the mean-squared error in a state estimation task using Gaussian-tuned point processes as sensors. This allows us to study the dynamics of the error of an optimal Bayesian decoder, providing insights into the limits obtainable in this task. This is done for Markovian and a class of non-Markovian Gaussian processes. We find that there is an optimal tuning width for which the error is minimized. This leads to a characterization of the optimal encoding for the setting as a function of the statistics of the stimulus, providing a mathematically sound primer for an ecological theory of sensory processing.

## 1 Introduction

Biological systems are constantly interacting with a dynamic, noisy environment, which they can only assess through noisy sensors. Models of Bayesian decision-making have been suggested to account for the functioning of biological systems in many areas [1, 2]. Here, we concentrate on the problem of Bayesian filtering of stochastic processes. There have been many studies on filtering of stimuli by biological systems [1, 2, 3], however, there are very few analytical results regarding the error of Bayesian filtering. We provide exact expressions for the evolution of the Mean Squared Error (MSE) of Bayesian filtering for a class of Gaussian processes. Results for expected errors of Gaussian processes had been sofar obtained only for the problem of *smoothing*, where predictions are not online but have to be made using past and future observations [4, 5].

The present work seeks to give an account of the error properties in Bayesian filtering of stochastic processes. We start by analysing the case of Markovian processes in section 2. We find a set of filtering equations from which we can derive a differential equation for the expected mean squared error. This provides a way to optimize the system parameters (the 'encoder') in order to minimize the error. We present an implicit equation to optimize the encoding scheme in the case of Poisson spike observations. We also provide a full stochastic model of the evolution of the error, which can

be solved analytically in a given interval. Useful approximations for the distribution of the error are also provided. In section 3 we show an application to optimal population coding in sensory neurons. In section 4 we extend the same framework to higher order processes, where we can control the smoothness by the order of the process. We finalize with a brief discussion. Our theoretical results contribute to the ongoing research on ecological theories in biological signal processing (e.g., [6]), which argue that performance of sensory systems can be enhanced by allowing sensors to adapt to the statistics of the environment. While an increasing amount of biological evidence has been accumulating for such theories (e.g., [7, 8, 9, 10, 11]) there has been little work providing exact analytic demonstration of its utility so far.

## 2 Bayesian Filtering for the Ornstein-Uhlenbeck Process

Consider the problem of estimating a dynamically evolving state in continuous time based on partial noisy observations. In classic approaches one assumes that the state is observed either continuously or at discrete times, leading to the celebrated Kalman filter and its extensions. We are concerned here with a setup of much interest in Neuroscience (as well as in Queueing theory) where the observations take the form of a a set of point processes. More concretely, let $X(t)$ be a stochastic process, and let $M$ 'sensory' processes be defined, each of which generates a Poisson point process with a time-dependent rate function $\lambda_m(X(t), t)$, $m = 1, 2, \ldots, M$. Such a stochastic process is often referred to as a doubly stochastic point process. In a neuroscience context $\lambda_m(\cdot)$ represents the tuning function of the $m$'th sensory cell. In order to maintain analytic tractability we focus in this work on a Gaussian form for $\lambda_m$, given by $\lambda_m(X(t), t) = \phi \exp\left[-(X(t) - \theta_m)^2 / 2\alpha(t)^2\right]$, where $\theta_m$ are the tuning function centers. We will assume the tuning function centers are equally spaced with spacing $\Delta\theta$, for simplicity, although this is not essential to our arguments.

Though the rate of observations for the individual processes depends on the instantaneous value of the process, it can be shown that under certain assumptions the total rate of observations (the rate by which observations by all processes are generated) is independent of the process. If we assume that the processes are independent and assume that the probability of the stimulus falling outside the range spanned by the tuning function centers is negligible, we obtain the total rate of observations

$$\lambda(t) \approx \sum_m \lambda_m(X(t), t) = \phi \sum_m \exp\left[-\frac{(X(t) - \theta_m)^2}{2\alpha^2(t)}\right] \approx \frac{\sqrt{2\pi}\phi\alpha(t)}{\Delta\theta}.$$

This approximation is discussed extensively in [12] and is seen to be very precise as long as $\alpha$ is of the same or of a larger order of magnitude as $\Delta\theta$. Denoting the set of observations generated by the sensory processes by $\boldsymbol{\xi} = \{(t_i, m_i, \Theta_i)\}$[1], we have the probability of a given set of observations $\boldsymbol{\xi}$ given a stimulus history $X_{[t_0, t]}$

$$P(\boldsymbol{\xi}|X_{[t_0, t]}) = e^{-\sum_m \int_{t_0}^{t_f} \lambda_m(X(t), t)dt} \prod_i \lambda_{m_i}(X(t_i), t_i) = e^{-\int_{t_0}^{t_f} \lambda(t)dt} \prod_i \lambda_{m_i}(X(t_i), t_i).$$

This defines the likelihood of the observations. Note that without the independence of the total rate from the stimulus, the likelihood would not be Gaussian due to the first term in the product. We need to evaluate the posterior probability $P(X(t)|\boldsymbol{\xi})$. We have

$$P(X(t)|\boldsymbol{\xi}) \propto P(X(t))P(\boldsymbol{\xi}|X(t)) = P(X(t)) \int d\mu(X_{[t_0, t)})P(\boldsymbol{\xi}|X_{[t_0, t)})P(X_{[t_0, t)}|X(t)).$$

The equations involved are Gaussian and evaluating them we obtain the usual Gaussian process regression equations (see [13] and [14, p. 17])

$$\mu(t, \boldsymbol{\xi}) = \sum_{i,j} K(t - t_i)C_{ij}^{-1}\Theta_j, \quad s(t, \boldsymbol{\xi}) = K(0) - \sum_{i,j} K(t - t_i)C_{ij}^{-1}K(t_j - t),^2 \quad (1)$$

where $K(t - t')$ is the auto-correlation function or kernel of the Gaussian process $X(t)$. This specifies the posterior distribution $P(X(t)|\boldsymbol{\xi}) = \mathcal{N}(\mu(t, \boldsymbol{\xi}), s(t, \boldsymbol{\xi}))$.

Our object of interest is the average mean squared error of the Bayesian estimator at a time $t$ based on past observations. This is the minimal mean-squared error of the optimal Bayesian estimator $\hat{X}(t; \boldsymbol{\xi}) = \langle X(t) \rangle_{X(t)|\boldsymbol{\xi}}$ with respect to a mean-squared error loss function. It is given by

$$MMSE(t) = \left\langle \left\langle (X(t) - \hat{X}(t; \boldsymbol{\xi}))^2 \right\rangle_{X(t)|\boldsymbol{\xi}} \right\rangle_{\boldsymbol{\xi}} = \left\langle \langle (X(t) - \mu(t; \boldsymbol{\xi}))^2 \rangle_{X(t)|\boldsymbol{\xi}} \right\rangle_{\boldsymbol{\xi}} = \langle s(t; \boldsymbol{\xi}) \rangle_{\boldsymbol{\xi}}.$$

Here we have written the averaging in the reverse of the usual order and have used $\hat{X}(t, \boldsymbol{\xi}) = \mu(t, \boldsymbol{\xi})$ in the second step. Note that the posterior variance is independent of the value of the observations, depending solely on the observation times. However the exact result is still intractable due to both the complex dependence of $s(t, \boldsymbol{\xi})$ on the observation times and the averaging over these. Note that so far the results hold for all kinds of Gaussian processes.

If we make a Markov assumption about the structure of the kernel $K(t - t')$ we are able to make statements about the evolution of the posterior variance between observations. This allows us to derive the differential Chapman-Kolmogorov equation [15] for the evolution of the posterior variance and then obtain the evolution of the MMSE. For the Ornstein-Uhlenbeck process $dX(t) = -\gamma X(t)dt + \eta dW(t)$ we have the kernel $k(\tau) = \frac{\eta^2}{2\gamma} e^{-\gamma|\tau|}$ and the differential equation for the evolution of the posterior variance between observations (see [16, p. 40] for example)

$$\frac{ds(t)}{dt} = -2\gamma s(t) + \eta^2. \tag{2}$$

When a new observation arrives, the distribution is updated through Bayes' rule. Using that $P(X(t)) = \mathcal{N}(\mu(t), s(t))$ and $P(\theta_i|X) \propto \mathcal{N}(\theta_i; X, \alpha^2(t))$, one can see that

$$P(X(t)|(t, \theta_i)) = \mathcal{N}\left( \frac{\alpha^2(t)\mu(t) + s(t)\theta_i}{\alpha^2(t) + s(t)}, \frac{\alpha^2(t)s(t)}{\alpha^2(t) + s(t)} \right). \tag{3}$$

Here, as before, the posterior variance is independent of the specific observation $\theta_i$, therefore we need only concentrate on the times of observations for purposes of modeling the posterior variance.

The evolution of the posterior variance is a Markov process which is driven by a deterministic drift, given in Eq. 2, and is also subject to discontinuous jumps at random times, which account for the observations, described by Eq. 3. This continuous time stochastic process is defined by a transition probability which in the time limit of infinitesimal time $dt \to 0$ is given by

$$P(s', t + dt|s, t) = (1 - \lambda(t)dt)\delta(s' - s + dt(2\gamma s - \eta^2)) + \lambda(t)dt\delta\left( s' - \frac{\alpha(t)^2 s}{\alpha(t)^2 + s} \right). \tag{4}$$

In the equation above, the first term accounts for the drift given in Eq. 2 and the second term accounts for the jumps given by Eq. 3. Using (4), and following a standard approach described in Gardiner [15, p. 47], we obtain a partial differential equation, the so-called differential Chapman-Kolmogorov equation for the *exact* time evolution of the marginal probability density $P(s, t)$

$$\frac{\partial P(s, t)}{\partial t} = \frac{\partial}{\partial s} \left[ (2\gamma s - \eta^2) P(s, t) \right] + \lambda \left( \frac{\alpha^2}{\alpha^2 - s} \right)^2 P\left( \frac{\alpha^2 s}{\alpha^2 - s}, t \right) - \lambda P(s, t). \tag{5}$$

This equation is, however, too complicated to be solved exactly in the general case. We can use it to derive the evolution of statistical averages by noting that $\frac{d\langle f(s) \rangle}{dt} = \int ds f(s) \frac{\partial P(s, t)}{\partial t}$. For $f(s) = s$ we obtain an exact equation for the evolution of the average error. Writing $\epsilon = \langle s \rangle$, we have

$$\frac{d\epsilon}{dt} = -2\gamma\epsilon + \eta^2 - \lambda(t) \left\langle \frac{s^2}{\alpha^2(t) + s} \right\rangle_{P(s,t)}. \tag{6}$$

## 2.1 Mean field approximation

We will now derive a good closed form *approximate equation* for the expected posterior variance $\epsilon = \langle s \rangle$ from (6). Note that the expectation of the nonlinear function on the right hand side is again intractable but can be approximated using a mean-field approximation of the type $\langle f(s) \rangle \approx f(\langle s \rangle)$. We obtain

$$\frac{d\epsilon_{mf}}{dt} = -2\gamma\epsilon_{mf} + \eta^2 - \lambda(t) \frac{\epsilon_{mf}^2}{\alpha(t)^2 + \epsilon_{mf}}. \tag{7}$$

This approximation works remarkably well, giving an excellent account of the equilibrium regime and of the relaxation of the error as can be seen in Fig. 2 for the case of population coding. We can also minimize the change in $\epsilon$ at each time step with respect to the sensor parameters $\alpha, \phi$ to find optimal values for them. The maximal observation rate $\phi$ is quite trivial, as an increase in $\phi$ increases the effect of observations linearly. Therefore without a cost associated to observations, there is no optimal value for $\phi$, since increasing it will always lead to lower values of $\epsilon$. Minimizing the derivative of $\epsilon$ with respect to $\alpha$ however, yields an implicit equation for the optimal value of $\alpha(t)$

$$\alpha_{opt}(t)^2 = \left\langle \frac{s^3}{(\alpha_{opt}(t)^2 + s)^2} \right\rangle_{P(s,t)} \Big/ \left\langle \frac{s^2}{(\alpha_{opt}(t)^2 + s)^2} \right\rangle_{P(s,t)} \tag{8}$$

Using again a mean-field approach, we obtain the simple result for the time-dependent tuning width $\alpha_{opt}^2(t) = \epsilon(t)$, so the square of the optimal tuning width is the average error of the current estimate of the process. This is interesting as it accounts for sharpening of the gaussian rates when the error is small and broadening when the error is large.

## 2.2 Exact results for the stationary distribution

We will now assume that both $\lambda$ and $\alpha$ are time independent so that the stochastic process converges to a stationary state described by $\frac{\partial P(s,t)}{\partial t} = 0$. To obtain information about this stationary solution it is useful to introduce the new variable $z = \eta^2/(\gamma s)$. The linear ODE 2 transforms into a nonlinear one $\dot{z}(t) = \gamma z(2 - z)$. This slight complication comes with a great simplification for the jump conditions. In the new variable this is simply $z' = z + \delta$ where $\delta = \eta^2/(\gamma \alpha^2)$ does not depend on $z$. Hence the differential Chapman-Kolmogorov equation (specialised to the stationary state) is simply

$$-\frac{d}{dz}\left[\gamma z(2 - z)P(z)\right] + \lambda P(z - \delta) - \lambda P(z) = 0 \tag{9}$$

Viewing $z$ as a temporal variable, we can treat Eq. 9 as a *delay differential equation* which depends on $p$ at *previous* values of $z$. If we knew $P(z)$ in an interval $z_0 - \delta \leq z < z_0$, Eq. 9 would however become a simple ordinary linear differential equation with a known inhomogeneity $P(z - \delta)$ in the interval $z_0 \leq z \leq z_0 + \delta$ which could be solved explicitly by numerical quadrature. Repeating this procedure would allow us to obtain $p(z)$ iteratively for all $z > 0$. A simple argument shows that $P(z) = 0$ for $z < 2$. Since jumps can only *increase* $z$ and since also $\dot{z}(t) > 0$ for $z < 2$, we find that in the stationary state, the interval $0 \leq z < 2$ will become depopulated. Hence, for $2 \leq z \leq 2 + \delta$ we have

$$-\frac{d}{dz}\left[\gamma z(2 - z)P(z)\right] = \lambda P(z)$$

which is solved by $P(z) \propto z^{-2}(1 - 2/z)^{-1 + \lambda/2\gamma}$ Transforming back to the original error variable $s$ yields

$$P_{eq}(s) \propto (\eta^2 - 2\gamma s)^{\frac{\lambda}{2\gamma} - 1}. \tag{10}$$

valid for $s \in \left[\frac{\alpha^2 \eta^2}{2\gamma\alpha^2 + \eta^2}, \frac{\eta^2}{2\gamma}\right]$. This is a very interesting result, as it shows a diverging behaviour in the equilibrium for values of $\lambda < 2\gamma$. This singularity can also be verified in the simulations. This solution gives us a good intuition about the coding properties of the system. When the average time between observations $\tau_{obs} = 1/\lambda$ is smaller than the relaxation time of the process' variance $\tau_{var} = 1/2\gamma$, the most probable value for the error will be the equilibrium variance of the observed process $\eta^2/2\gamma$. Note however that the expected error is always smaller than $\eta^2/2\gamma$. When $\lambda = 2\gamma$ we observe a transition and the most likely error becomes smaller. It was not possible to give closed form analytical expressions for $p(z)$ in the following intervals because the integrals are not analytically tractable. We can, however, solve (10) numerically obtaining great agreement with the simulated histograms. For very small values of $\delta$, the numerical integration becomes less reliable, as the valid intervals become increasingly small, requiring a very small integration step. This can be seen in Fig. 1.

We can get *asymptotic* expressions for $P(z)$ when parameters are such that the relative fluctuations of $z$ are small. This is expected to hold for small jumps $\delta$ (when the system is trivially almost deterministic) and/or for large jump rates $\lambda$, when the density of jumps is so large that relative fluctuations are small. Using again a simple mean field argument as before shows that in such

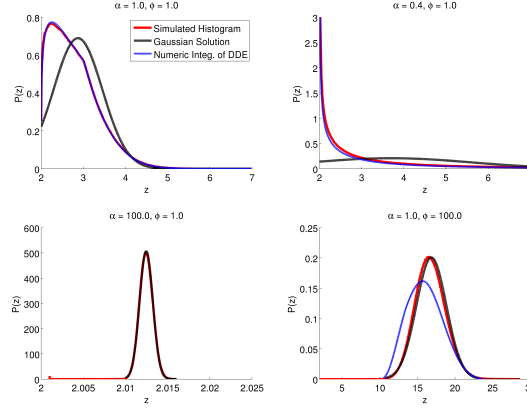

Figure 1: Comparison of the different regimes for the equilibrium distribution. Top left we can see $\alpha = \phi = 1$. Note that neither solutions cover all of the range of the distribution, although the exact solution captures the behaviour very well in the low $z$ region. Top right we can see the low $\alpha$ regime. Note that the exact solution accounts for the distribution on most of the range of the distribution. In the bottom we see the cases where the Gaussian approximation excels. Both large $\alpha$ and $\phi$ result in an approximately Gaussian distribution, as we have derived above. The blue line (exact solution) is hardly discernible from the red line (histogram) in the small $\alpha$ case, as is the black line (Gaussian approximation) in the large $\alpha$ or $\phi$ case.

situations we find that in equilibrium $z$ should be close to $z^* = 1 + \sqrt{1 + \lambda\delta/\gamma}$. For both small $\delta$ and/or large $\lambda$, for $z$ close to $z^*$ we have $\delta \ll z^*$ and we can expand $p(z - \delta)$ in a Taylor series to second order in $\delta$. Linearising also the drift $\gamma z(2 - z)$ around $z^*$ yields a Fokker-Planck equation which is equivalent to a simple diffusion process (of the Ornstein-Uhlenbeck type) which is solved by the Gaussian density $P(z) = \mathcal{N}\left(1 + \sqrt{1 + \lambda\delta/\gamma}, \frac{\lambda\delta^2}{(4\gamma\sqrt{1+\lambda\delta/\gamma})}\right)$. In Fig. 1 we present the different approximations compared to the simulated histograms of the posterior variance.

We present results for the specific choice $\eta = \gamma = 1$. Note however, that through a scaling of parameters $\alpha' = \alpha\eta/\sqrt{\gamma}$ and $\phi' = \phi\gamma$ we can obtain the MMSE for any value of the four parameters with the values for $\eta = \gamma = 1$. In this way, rescaling the parameters, we can obtain the MMSE for any values of $\eta$, $\gamma$, $\alpha$ and $\phi$.

## 3 Optimal Population Coding

As an application we look into the problem of neural population coding of dynamic stimuli (see [13]). We model the spiking of neurons as doubly stochastic Poisson processes driven by the stimulus $X(t)$, that is the probability of a given neuron firing a spike in a given interval $[t, t + dt]$ is given by

$$P_t(\text{spike}_m|X(t)) = \phi e^{-\frac{(X(t) - \theta_m)^2}{2\alpha(t)^2}} dt, \quad \text{and} \quad P_t(\text{spike}|X(t)) \approx \frac{\sqrt{2\pi}\phi\alpha(t)dt}{\Delta\theta} = \lambda(t)dt.$$

Under these assumptions, the inference from a spike train is equivalent to that on observations of data, and the MMSE follows the differential Eq. 6. Again, the fact that the posterior variance depends solely on the spike times allows us to substitute the spiking processes for each neuron with one spiking process for the whole population, simplifying greatly our calculations. We compare the framework derived with the dynamic population coding presented in [13] in Fig. 2.

We have calculated the MMSE for a range of values for $\alpha$ and $\phi$ to obtain the dependence of the MMSE on these parameters. In Fig. 3 we show the mean-field treatment of Eq. 6 as well as simulations of the dynamics given by Eq. 4. The mean-field approximation works remarkably well, yielding a relative error smaller than $2\%$ throughout the range of parameters. The approximate and simulated error maps are virtually indistinguishable. As can be seen in Fig. 2 the mean-field approximation also works very well to reflect the dynamics of the error.

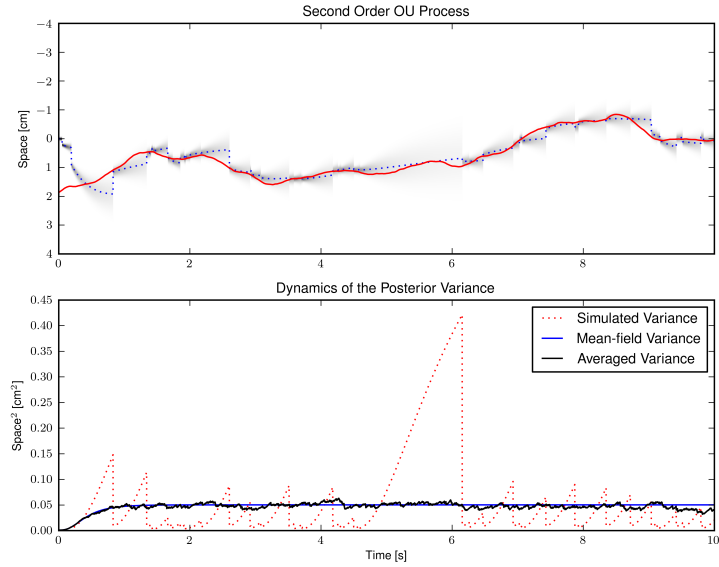

Figure 2: Neural coding of an second-order Markov process as described in the text. Top figure shows the process overlayed with posterior mean and confidence intervals. The bottom plot shows the posterior variance of one sample run in black, the average over a thousand runs in blue and the mean-field dynamics in red. Code modified from [13]

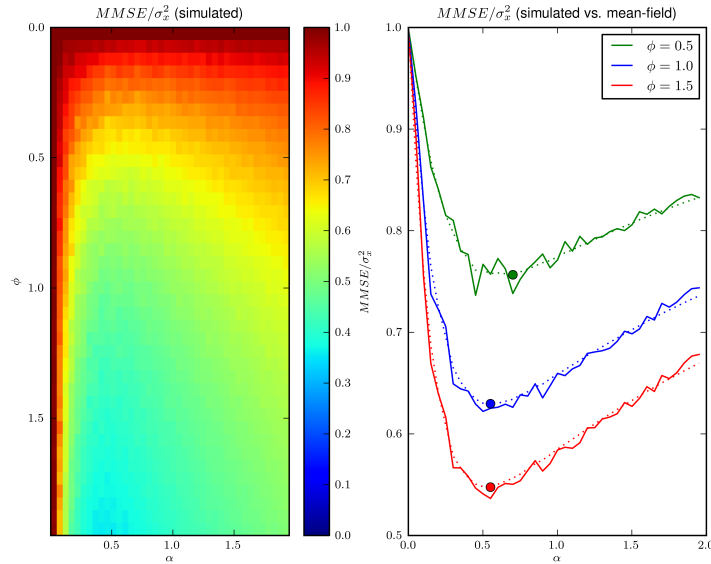

Figure 3: MMSE for the Ornstein-Uhlenbeck process. On the left we have the average MMSE obtained by the simulation and on the right the value of the MMSE as a function of $\alpha$ for a few values of $\phi$ in the mean-field approximation. The dots are the minima for the mean-field and the dotted curves are mean-field values for the same $\phi$. The mean-field leads to a very good approximation, and the optimal $\alpha$ for the approximation is a good estimator for the optimal $\alpha$ in the simulation.

## 4 Filtering Smoother Processes

To study the filtering of smoother processes we will look at higher-order Markov processes. We do so by considering a multidimensional stochastic process which is Markovian if we consider all of the components, but restrict ourselves to one component, which will then exhibit a non-Markovian structure. This is done by an extension to the Ornstein-Uhlenbeck process frequently used in Gaussian process literature, whose correlation structure is given by the Matern kernel (see below). We have to work with the covariance matrix of the system, since its elements' dynamics are coupled. Thus, Eq. 6 will be replaced by a matrix equation, to which we then apply the same treatment.

We consider a $p$-th order stochastic process such as $a_{p+1}X^{(p)}(t) + a_p X^{(p-1)}(t) + \cdots + a_1 X(t) = \eta Z(t)$, where $Z(t)$ is white Gaussian noise with covariance $\delta(t - t')$ and $X^{(n)}(t)$ denotes the $n$-th derivative of $X(t)$. Writing the proper Ito stochastic differential equations we obtain a set of $p - 1$ first order differential equations and a single first order stochastic differential equation,

$$\dot{X}_1 = X_2, \dot{X}_2 = X_3, \ldots, \dot{X}_{p-1} = X_p, \quad a_{p+1}dX_p = -\sum_{i=1}^{p} a_i X_i dt + \eta dW_t,$$

where $W_t$ is the Wiener process. Choosing $a_k = \binom{p}{k-1}\gamma^{p+1-k}$ which yields processes $X_1(t)$ with an autocorrelation function given by the Matern kernel

$$k(\tau; \nu, \gamma) = \frac{\eta^2 2^{-\nu}}{\sqrt{\pi}\Gamma(\nu + 1/2)\gamma^\nu} \left(\gamma\tau\right)^\nu K_\nu\left(\gamma\tau\right),$$

where $\nu + \frac{1}{2} = p$, $K_\nu(x)$ is the modified Bessel function of the second kind and $\gamma$ is the parameter determining the characteristic time of the kernel. Note that the one-dimensional Ornstein-Uhlenbeck process is a special case of this with $p = 1, \nu = 1/2$. We can control the smoothness of the process $X_1(t)$ with the parameter $\nu$, increasing it yields successively smoother processes (see supplementary information).

We can express this as a multidimensional stochastic process by choosing $\Gamma_{i,j} = -\delta_{i,j-1} + \delta_{i,p}a_j$ and $\Sigma^{1/2}_{i,j} = \delta_{i,p}\delta_{j,p}\eta$, where $\delta_{i,j}$ is the Kronecker delta. We then have the Ito stochastic differential equation

$$d\vec{X}(t) = -\Gamma\vec{X}(t)dt + \Sigma^{1/2}d\vec{W} \tag{11}$$

for $\vec{X}(t)^T = (X_1(t), X_2(t), \ldots, X_p(t))$. The covariance matrix then evolves according to (see [16, p. 40])

$$\frac{d\sigma}{dt} = -\Gamma\sigma - \sigma\Gamma^T + \Sigma. \tag{12}$$

This can be solved using the solution of the homogeneous equation $\Sigma(t) = \exp[-t\Gamma]\exp[-t\Gamma^t]$ and the solution to the inhomogeous equation given by the equilibrium solution.

We assume that only the component $X_1$ is observed, that is, the rate of observations only depends on that component. We have then $P(X_1, X_{2:p}|\text{obs}) \propto P(\text{obs}|X_1)P(X_1, X_{2:p})$. Note that the precision matrix (the inverse of the covariance matrix) will be updated simply by adding the likelihood term $1/\alpha(t)^2$ to the first diagonal element. Using the block matrix inversion theorem we obtain the new covariance matrix

$$\sigma'_{i,j} = \sigma_{i,j} - \frac{\sigma_{1,i}\sigma_{1,j}}{\alpha^2 + \sigma_{1,1}}. \tag{13}$$

Putting equations 12 and 13 together we obtain the differential Chapman-Kolmogorov equation for the evolution of the probability of the covariance matrix. With this we obtain the differential equation for the average posterior covariance matrix

$$\frac{d\langle\sigma_{i,j}\rangle}{dt} = \langle\sigma_{i+1,j}\rangle + \langle\sigma_{i,j+1}\rangle - \sum_l \left(\delta_{i,p}a_l\langle\sigma_{i,l}\rangle + \delta_{j,p}a_l\langle\sigma_{j,l}\rangle\right) - \lambda(t)\left\langle\frac{\sigma_{1,i}\sigma_{1,j}}{\alpha(t)^2 + \sigma_{1,1}}\right\rangle + \eta^2\delta_{i,n}\delta_{j,n}, \tag{14}$$

where we abuse the notation by using that $\sigma_{i,j} = 0$, if $i > p$ or $j > p$. These can be solved in the mean-field approximation to obtain an approximation for the covariance matrix. We also note that one can derive a recursion scheme to express all of the elements as functions of the first row of covariances $\sigma_{1,1:p}$. With these expressions we can then use the equilibrium conditions for $\frac{d\langle\sigma_{i,i}\rangle}{dt}$

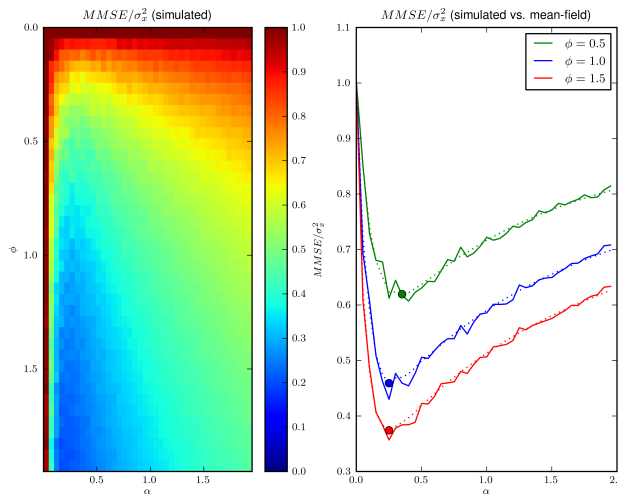

Figure 4: MMSE for a second-order stochastic process. On the left is the color map of the first diagonal element of the covariance matrix for the $\nu = 3/2$ case, corresponding to the variance of the observed stimulus variable and on the right, the same element as a function of $\alpha$ for a few values of $\phi$. The overall dependence of the error on $\alpha$ and $\phi$ is strikingly similar to the OU process, with lower values of the MMSE, however. This is due to the smoothness of the process, making it more predictable. In red we show the MMSE for the simulated equilibrium variance for comparison. Though the mean-field approximation is not as good as in the OU case, the relative error of it still falls below 18% throughout the range of parameters studied.

to solve for the equilibrium value of $\langle \sigma_{i,j} \rangle$. We provide results for the case $p = 2$, $\nu = 3/2$. The equilibrium MMSE is shown in Fig. 4 on the left and in Fig. 4 on the right we show the dependence on $\alpha$ of the MMSE. The dependence of the error on the parameters resembles strongly that of the Ornstein-Uhlenbeck process, showing a finite optimal value of $\alpha$ which minimizes the error given $\phi$. This becomes less pronounced as we go to very low firing rates. Note that for the second-order process the MMSE relative to the variance of the observed process ($MMSE/K(0)$) drops to lower values than in the Ornstein-Uhlenbeck process, leading to a better state estimation. We expect that the error will become increasingly smaller for higher-order processes.

# 5 Discussion

We have shown that the dynamics of Bayesian state estimation error for Markovian processes can be modelled by a simple dynamic system. This provides insight into generalization properties of Gaussian process inference in an online, causal setting, where previous generalization error calculations [4, 5] for Gaussian processes do not apply. In the context of filtering the usual generalization error calculations do not apply. Furthermore, we have demonstrated that a simple mean-field approximation succesfully captures the dynamics of the average error of the described inference framework. This was shown in detail for the case of Ornstein-Uhlenbeck processes, and for a class of higher-order Markov processes.

One key feature we were able to verify is the existence of an optimal tuning width for Gaussian-tuned Poisson processes which minimizes the MMSE, as has been verified elsewhere for static stimuli ([17, 12, 18]). This result is robust to the inclusion of coloured noise, as we have shown by modelling a second order process.

Future research could concentrate in generalizing the presented framework towards more realistic spike generation models, such as integrate-and-fire neurons. The generalization to broader classes of stimuli would be of great interest as well. These results provide a promising first step towards a mathematical theory of ecologically grounded sensory processing.

# 6    Acknowledgements

The work of Alex Susemihl was supported by the DFG Research Training Group GRK1589/1. The work of Ron Meir was partially supported by grant No. 665/08 from the Israel Science Foundation.

## Footnotes

[1]Here the time $t_i$ denotes the time of the $i$-th observation, $m_i$ gives the identity of the sensor making the observation and $\Theta_i = \theta_{m_i}$ is the mean of the Gaussian rate function.

[2]$C_{ij}(\boldsymbol{\xi}) = K(t_i - t_j) + \delta_{ij}\alpha(t_i)^2$

# References

[1] Tetsuya J. Kobayashi. Implementation of dynamic bayesian decision making by intracellular kinetics. *Phys. Rev. Lett.*, 104(22):228104, Jun 2010.

[2] Jean-Pascal Pfister, Peter Dayan, and Mate Lengyel. Know thy neighbour: A normative theory of synaptic depression. In Y. Bengio, D. Schuurmans, J. Lafferty, C. K. I. Williams, and A. Culotta, editors, *Advances in Neural Information Processing Systems 22*, pages 1464–1472. 2009.

[3] Omer Bobrowski, Ron Meir, Shy Shoham, and Yonina C. Eldar. A neural network implementing optimal state estimation based on dynamic spike train decoding. In *Neural Information Processing Systems*, 2007.

[4] Dorthe Malzahn and Manfred Opper. A statistical physics approach for the analysis of machine learning algorithms on real data. *Journal of Statistical Mechanics: Theory and Experiment*, 2005(11):P11001, 2005.

[5] P. Sollich and A. Halees. Learning curves for gaussian process regression: Approximations and bounds. *Neural Computation*, 14(6):1393–1428, 2002.

[6] J Atick and A.N. Redlich. Could information theory provide an ecological theory of sensory processing? *Network: Computation in Neural Systems*, 5:213–251, 1992.

[7] M.W. Pettet and C.D. Gilbert. Dynamic changes in receptive-field size in cat primary visual cortex. *Proceedings of the National Academy of Sciences*, 89(17):8366–8370, 1992.

[8] N. Brenner, W. Bialek, and R. de Ruyter van Steveninck. Adaptive rescaling maximizes information transmission. *Neuron*, 26(3):695–702, 2000.

[9] V. Dragoi, J. Sharma, and M. Sur. Adaptation-induced plasticity of orientation tuning in adult visual cortex. *Neuron*, 28(1):287–298, 2000.

[10] I. Dean, B.L. Robinson, N.S. Harper, and D. McAlpine. Rapid neural adaptation to sound level statistics. *Journal of Neuroscience*, 28(25):6430–6438, 2008.

[11] T. Hosoya, S.A. Baccus, and M. Meister. Dynamic predictive coding by the retina. *Nature*, 436(7047):71–77, 2005.

[12] Steve Yaeli and Ron Meir. Error-based analysis of optimal tuning functions explains phenomena observed in sensory neurons. *Frontiers in Computational Neuroscience*, 5(0):12, 2010.

[13] Quentin J. M. Huys, Richard S. Zemel, Rama Natarajan, and Peter Dayan. Fast population coding. *Neural Computation*, 19(2):404–441, 2007.

[14] C.E. Rasmussen and C.K.I. Williams. *Gaussian Processes for Machine Learning*. The MIT Press, 55 Hayward Street, Cambridge, MA 02142, 2006.

[15] C.W. Gardiner. *Stochastic Methods: A Handbook for the Natural and Social Sciences*, volume 13 of *Springer Serier in Synergetics*. Springer, Berlin Heidelberg, fourth edition, 2009.

[16] Hannes Risken. *The Fokker-Planck Equation: Methods of Solutions and Applications*, volume 18 of *Springer Series in Synergetics*. Springer, Berlin Heidelberg, second ed. 1989. third printing edition, 1996.

[17] M. Bethge, D. Rotermund, and K. Pawelzik. Optimal short-term population coding: When fisher information fails. *Neural Computation*, 14(10):2317–2351, 2002.

[18] Philipp Berens, Alexander S. Ecker, Sebastian Gerwinn, Andreas S. Tolias, and Matthias Bethge. Reassessing optimal neural population codes with neurometric functions. *Proceedings of the National Academy of Sciences*, 108(11):4423–4428, 2011.

